# A Divide-and-Conquer Procedure for Sparse Inverse Covariance Estimation

**Cho-Jui Hsieh**
Dept. of Computer Science
University of Texas, Austin
cjhsieh@cs.utexas.edu

**Inderjit S. Dhillon**
Dept. of Computer Science
University of Texas, Austin
inderjit@cs.utexas.edu

**Pradeep Ravikumar**
Dept. of Computer Science
University of Texas
pradeepr@cs.utexas.edu

**Arindam Banerjee**
Dept. of Computer Science & Engineering
University of Minnesota, Twin Cities
banerjee@cs.umn.edu

## Abstract

We consider the composite log-determinant optimization problem, arising from the $\ell_1$ regularized Gaussian maximum likelihood estimator of a sparse inverse covariance matrix, in a high-dimensional setting with a very large number of variables. Recent work has shown this estimator to have strong *statistical guarantees* in recovering the true structure of the sparse inverse covariance matrix, or alternatively the underlying graph structure of the corresponding Gaussian Markov Random Field, even in very high-dimensional regimes with a limited number of samples. In this paper, we are concerned with the *computational* cost in solving the above optimization problem. Our proposed algorithm partitions the problem into smaller sub-problems, and uses the solutions of the sub-problems to build a good approximation for the original problem. Our key idea for the *divide* step to obtain a sub-problem partition is as follows: we first derive a tractable bound on the quality of the approximate solution obtained from solving the corresponding sub-divided problems. Based on this bound, we propose a clustering algorithm that attempts to minimize this bound, in order to find effective partitions of the variables. For the *conquer* step, we use the approximate solution, i.e., solution resulting from solving the sub-problems, as an initial point to solve the original problem, and thereby achieve a much faster computational procedure.

## 1 Introduction

Let $\{\boldsymbol{x}_1, \boldsymbol{x}_2, \ldots, \boldsymbol{x}_n\}$ be $n$ sample points drawn from a $p$-dimensional Gaussian distribution $\mathcal{N}(\mu, \Sigma)$, also known as a Gaussian Markov Random Field (GMRF), where each $\boldsymbol{x}_i$ is a $p$-dimensional vector. An important problem is that of recovering the covariance matrix, or its inverse, given the samples in a high-dimensional regime where $n \ll p$, and $p$ could number in the tens of thousands. In such settings, the computational efficiency of any estimator becomes very important.

A popular approach for such high-dimensional inverse covariance matrix estimation is to impose the structure of sparsity on the inverse covariance matrix (which can be shown to encourage conditional independences among the Gaussian variables), and to solve the following $\ell_1$ regularized maximum likelihood problem:

$$\arg\min_{\Theta \succ 0}\{-\log\det\Theta + \operatorname{tr}(S\Theta) + \lambda\|\Theta\|_1\} = \arg\min_{\Theta \succ 0} f(\Theta), \tag{1}$$

where $S = \frac{1}{n}\sum_{i=1}^{n}(\boldsymbol{x}_i - \tilde{\mu})(\boldsymbol{x}_i - \tilde{\mu})^T$ is the sample covariance matrix and $\tilde{\mu} = \frac{1}{n}\sum_{i=1}^{n}\boldsymbol{x}_i$ is the sample mean. The key focus in this paper is on developing computationally efficient methods to solve this composite log-determinant optimization problem.

Due in part to its importance, many optimization methods [4, 1, 9, 7, 6] have been developed in recent years for solving (1). However, these methods have a computational complexity of at least $O(p^3)$ (typically this is the complexity *per iteration*). It is therefore hard to scale these procedures to problems with tens of thousands of variables. For instance, in a climate application, if we are modeling a GMRF over random variables corresponding to each Earth grid point, the number of nodes can easily number in the tens of thousands. For this data, a recently proposed state-of-the-art method QUIC [6], that uses a Newton-like method to solve (1), for instance takes more than 10 hours to converge.

A natural strategy when the computational complexity of a procedure scales poorly with the problem size is a divide and conquer strategy: Given a partition of the set of nodes, we can first solve the $\ell_1$ regularized MLE over the sub-problems invidually, and than in the second step, aggregate the solutions together to get $\bar{\Theta}$. But how do we come up with a suitable partition? The main contribution of this paper is to provide a principled answer to this question. As we show, our resulting divide and conquer procedure produces overwhelming improvements in computational efficiency.

Interestingly, [8] recently proposed a decomposition-based method for GMRFs. They first observe the following useful property of the composite log-determinant optimization problem in (1): if we threshold the off-diagonal elements of the sample covariance matrix $S$, and the resulting thresholded matrix is block-diagonal, then the corresponding inverse covariance matrix has the same block-diagonal sparsity structure as well. Using this property, they decomposed the problem along these block-diagonal components and solved these separately, thus achieving a sharp computational gain. A major drawback to this approach of [8] however is that often the decomposition of the thresholded sample covariance matrix can be very unbalanced — indeed, in many of our real-life examples, we found that the decomposition resulted in one giant component and several very small components. In these cases, the approach in [8] is only a bit faster than directly solving the entire problem.

In this paper, we propose a different strategy based on the following simple idea. Suppose we are given a particular partitioning, and solve the sub-problems specified by the partition components. The resulting decomposed estimator $\bar{\Theta}$ clearly need not be equal to $\ell_1$ regularized MLE (1). However, can we use bounds on the deviation to propose a clustering criterion? We first derive a bound on $\|\bar{\Theta} - \Theta^*\|_F$ based on the off-diagonal error of the partition. Based on this bound, we propose a normalized-cut spectral clustering algorithm to minimize the off-diagonal error, which is able to find a balanced partition such that $\bar{\Theta}$ is very close to $\Theta^*$. Interestingly, we show that this clustering criterion can also be motivated as leveraging a property more general than that in [8] of the $\ell_1$ regularized MLE (1). In the "conquering" step, we then use $\bar{\Theta}$ to initialize an iterative solver for the original problem (1). As we show, the resulting algorithm is much faster than other state-of-the-art methods. For example, our algorithm can achieve an accurate solution for the climate data problem in 1 hour, whereas directly solving it takes 10 hours.

In section 2, we outline the standard skeleton of a divide and conquer framework for GMRF estimation. The key step in such a framework is to come up with a suitable and efficient clustering criterion. In the next section 3, we then outline our clustering criteria. Finally, in Section 4 we show that in practice, our method achieves impressive improvements in computational efficiency.

## 2 The Proposed Divide and Conquer Framework

We first set up some notation. In this paper, we will consider each $p \times p$ matrix $X$ as an adjacency matrix, where $\mathcal{V} = \{1, \ldots, p\}$ is the node set, $X_{ij}$ is the weighted link between node $i$ and node $j$. We will use $\{\mathcal{V}_c\}_{c=1}^k$ to denote a disjoint partitioning of the node set $\mathcal{V}$, and each $\mathcal{V}_c$ will be called a partition or a cluster.

Given a partition $\{\mathcal{V}_c\}_{c=1}^k$, our divide and conquer algorithm first solves GMRF for all node partitions to get the inverse covariance matrices $\{\Theta^{(c)}\}_{c=1}^k$, and then uses the following matrix

$$\bar{\Theta} = \begin{bmatrix} \Theta^{(1)} & 0 & \ldots & 0 \\ 0 & \Theta^{(2)} & \ldots & 0 \\ \vdots & \vdots & \vdots & \vdots \\ 0 & 0 & 0 & \Theta^{(k)} \end{bmatrix}, \tag{2}$$

to initialize the solver for the whole GMRF. In this paper we use $X^{(c)}$ to denote the submatrix $X_{\mathcal{V}_c, \mathcal{V}_c}$ for any matrix $X$. Notice that in our framework any sparse inverse covariance solver can

be used, however, in this paper we will focus on using the state-of-the-art method QUIC [6] as the base solver, which was shown to have super-linear convergence when close to the solution. Using a better starting point enables QUIC to more quickly reach this region of super-linear convergence, as we will show later in our experiments.

The skeleton of the divide and conquer framework is quite simple and is summarized in Algorithm 1. In order that Algorithm 1 be efficient, we require that $\bar{\Theta}$ defined in (2) should be close to the optimal solution of the original problem $\Theta^*$. In the following, we will derive a bound for $\|\Theta^* - \bar{\Theta}\|_F$. Based on this bound, we propose a spectral clustering algorithm to find an effective partitioning of the nodes.

---

**Algorithm 1**: Divide and Conquer method for Sparse Inverse Covariance Estimation

**Input** : Empirical covariance matrix $S$, scalar $\lambda$
**Output**: $\Theta^*$, the solution of (1)
1 Obtain a partition of the nodes $\{\mathcal{V}_c\}_{c=1}^k$ ;
2 **for** $c = 1, \ldots, k$ **do**
3      Solve (1) on $S^{(c)}$ and subset of variables in $\mathcal{V}_c$ to get $\Theta^{(c)}$;
4 **end**
5 Form $\bar{\Theta}$ by $\Theta^{(1)}, \Theta^{(2)}, \ldots, \Theta^{(k)}$ as in (2) ;
6 Use $\bar{\Theta}$ as an initial point to solve the whole problem (1) ;

---

### 2.1 Hierarchical Divide and Conquer Algorithm

Assume we conduct a $k$-way clustering, then the initial time for solving sub-problems is at least $O(k(p/k)^3) = O(p^3/k^2)$ where $p$ denotes the dimensionality, When we consider $k = 2$, the divide and conquer algorithm can be at most 4 times faster than the original one. One can increase $k$, however, a larger $k$ entails a worse initial point for training the whole problem.

Based on this observation, we consider the hierarchical version of our divide-and-conquer algorithm. For solving subproblems we can again apply a divide and conquer algorithm. In this way, the initial time can be much less than $O(p^3/k^2)$ if we use divide and conquer algorithm hierarchically for each level. In the experiments, we will see that this hierarchical method can further improve the performance of the divide-and-conquer algorithm.

## 3 Main Results: Clustering Criteria for GMRF

This section outlines the main contribution of this paper; in coming up with suitable efficient clustering criteria for use within the divide and framework structure in the previous section.

### 3.1 Bounding the distance between $\Theta^*$ and $\bar{\Theta}$

To start, we discuss the following result from [8], which we reproduce using the notation in this paper for convenience. Specifically, [8] shows that when all the between cluster edges in $S$ have absolute values smaller than $\lambda$, $\Theta^*$ will have a block-diagonal structure.

**Theorem 1** ([8]). *For any $\lambda > 0$ and a given partition $\{\mathcal{V}_c\}_{c=1}^k$, if $|S_{ij}| \leq \lambda$ for all $i, j$ in different partitions, then $\Theta^* = \bar{\Theta}$, where $\Theta^*$ is the optimal solution of* (1) *and $\bar{\Theta}$ is as defined in* (2).

As a consequence, if a partition $\{\mathcal{V}_c\}_{c=1}^k$ satisfies the assumption of Theorem 1, $\bar{\Theta}$ and $\Theta^*$ will be the same, and the last step of Algorithm 1 is not needed anymore. Therefore the result in [8] may be viewed as a special case of our Divide-and-Conquer Algorithm 1.

However, in most real examples, a perfect partitioning as in Theorem 1 does not exist, which motivates a divide and conquer framework that does not need as stringent assumptions as in Theorem 1. To allow a more general relationship between $\Theta^*$ and $\bar{\Theta}$, we first prove a similar property for the following generalized inverse covariance problem:

$$\Theta^* = \arg\min_{\Theta \succ 0} \{-\log\det\Theta + \operatorname{tr}(S\Theta) + \sum_{i,j} \Lambda_{ij}|\Theta_{ij}|\} = \arg\min_{\Theta \succ 0} f_\Lambda(\Theta). \tag{3}$$

In the following, we use $\mathbf{1}_\lambda$ to denote a matrix with all elements equal to $\lambda$. Therefore (1) is a special case of (3) with $\Lambda = \mathbf{1}_\lambda$. In (3), the regularization parameter $\Lambda$ is a $p \times p$ matrix, where each element corresponds to a weighted regularization of each element of $\Theta$. We can then prove the following theorem, as a generalization of Theorem 1.

**Theorem 2.** *For any matrix regularization parameter $\Lambda$ ($\Lambda_{ij} > 0 \; \forall i, j$) and a given partition $\{\mathcal{V}_c\}_{c=1}^k$, if $|S_{ij}| \leq \Lambda_{ij}$ for all $i, j$ in different partitions, then the solution of (3) will be the block diagonal matrix $\bar{\Theta}$ defined in (2), where $\Theta^{(c)}$ is the solution for (3) with sample covariance $S^{(c)}$ and regularization parameter $\Lambda^{(c)}$.*

*Proof.* Consider the dual problem of (3):

$$\max_{W \succ 0} \; \log \det W \;\; \text{s.t.} \;\; |W_{ij} - S_{ij}| \leq \Lambda_{ij} \;\; \forall i, j, \tag{4}$$

based on the condition stated in the theorem, we can easily verify $\bar{W} = \bar{\Theta}^{-1}$ is a feasible solution of (4) with the objective function value $\sum_{c=1}^k \log \det \bar{W}^{(c)}$. To show that $\bar{W}$ is the optimal solution of (4), we consider an arbitrary feasible solution $\hat{W}$. From Fischer's inequality [2], $\det \hat{W} \leq \prod_{c=1}^k \det \hat{W}^{(c)}$ for $\hat{W} \succ 0$. Since $\bar{W}^{(c)}$ is the optimizer of the $c$-th block, $\det \bar{W}^{(c)} \geq \det \hat{W}^{(c)}$ for all $c$, which implies $\log \det \hat{W} \leq \log \det \bar{W}$. Therefore $\bar{\Theta}$ is the primal optimal solution. □

Next we apply Theorem 2 to develop a decomposition method. Assume our goal is to solve (1) and we have clusters $\{\mathcal{V}_c\}_{c=1}^k$ which may not satisfy the assumption in Theorem 1. We start by choosing a matrix regularization weight $\bar{\Lambda}$ such that

$$\bar{\Lambda}_{ij} = \begin{cases} \lambda & \text{if } i, j \text{ are in the same cluster,} \\ \max(|S_{ij}|, \lambda) & \text{if } i, j \text{ are in different clusters.} \end{cases} \tag{5}$$

Now consider the generalized inverse covariance problem (3) with this specified $\bar{\Lambda}$. By construction, the assumption in Theorem 2 holds for $\bar{\Lambda}$, so we can decompose this problem into $k$ sub-problems; for each cluster $c \in \{1, \dots, k\}$, the subproblem has the following form:

$$\Theta^{(c)} = \arg \min_{\Theta \succ 0} \{ -\log \det \Theta + \text{tr}(S^{(c)} \Theta) + \lambda \|\Theta\|_1 \},$$

where $S^{(c)}$ is the sample covariance matrix of cluster $c$. Therefore, $\bar{\Theta}$ is the optimal solution of problem (3) with $\bar{\Lambda}$ as the regularization parameter.

Based on this observation, we will now provide another view of our divide and conquer algorithm as follows. Considering the dual problem of the sparse inverse covariance estimation with the weighted regularization defined in (4), Algorithm 1 can be seen to solve (4) with $\Lambda = \bar{\Lambda}$ defined in (5) to get the initial point $\bar{W}$, and then solve (4) with $\Lambda = \mathbf{1}_\lambda$ for all elements. Therefore we initially solve the problem with looser bounded constraints to get an initial guess, and then solve the problem with tighter constraints. Intuitively, if the relaxed constraints $\bar{\Lambda}$ are close to the real constraint $\mathbf{1}_\lambda$, the solutions $\bar{W}$ and $W^*$ will be close to each other. So in the following we derive a bound based on this observation.

For convenience, we use $P^\lambda$ to denote the original dual problem (4) with $\Lambda = \mathbf{1}_\lambda$, and $P^{\bar{\Lambda}}$ to denote the relaxed dual problem with different edge weights across edges as defined in (5). Based on the above discussions, $W^* = (\Theta^*)^{-1}$ is the solution of $P^\lambda$ and $\bar{W} = \bar{\Theta}^{-1}$ is the solution of $P^{\bar{\Lambda}}$. We define $E$ as the following matrix:

$$E_{ij} = \begin{cases} 0 & \text{if } i, j \text{ are in the same cluster,} \\ \max(|S_{ij}| - \lambda, 0) & \text{otherwise.} \end{cases} \tag{6}$$

If $E = 0$, all the off-diagonal elements are below the threshold $\lambda$, so $W^* = \bar{W}$ by Theorem 2. In the following we consider a more interesting case where $E \neq 0$. In this case $\|E\|_F$ measures how much the off-diagonal elements exceed the threshold $\lambda$, and a good clustering algorithm should be able to find a partition to minimize $\|E\|_F$. In the following theorem we show that $\|W^* - \bar{W}\|_F$ can be bounded by $\|E\|_F$, therefore $\|\Theta^* - \bar{\Theta}\|_F$ can also be bounded by $\|E\|_F$:

**Theorem 3.** *If there exists a $\gamma > 0$ such that $\|E\|_2 \leq (1-\gamma)\frac{1}{\|\bar{W}\|_2}$, then*

$$\|W^* - \bar{W}\|_F < \frac{p \max(\sigma_{max}(\bar{W}), \sigma_{max}(W^*))}{\gamma \sigma_{min}(\bar{W})} \|E\|_F, \tag{7}$$

$$\|\Theta^* - \bar{\Theta}\|_F \leq \frac{p \max(\sigma_{max}(\bar{\Theta}), \sigma_{max}(\Theta^*))^2 \sigma_{max}(\bar{\Theta})}{\gamma \min(\sigma_{min}(\Theta^*), \sigma_{min}(\bar{\Theta}))} \|E\|_F, \tag{8}$$

*where $\sigma_{min}(\cdot), \sigma_{max}(\cdot)$ denote the minimum/maximum singular values.*

*Proof.* To prove Theorem 3, we need the following Lemma, which is proved in the Appendix:

**Lemma 1.** *If $A$ is a positive definite matrix and there exists a $\gamma > 0$ such that $\|A^{-1}B\|_2 \leq 1 - \gamma$, then*

$$\log \det(A + B) \geq \log \det A - p/(\gamma \sigma_{min}(A))\|B\|_F. \tag{9}$$

Since $P^{\bar{\lambda}}$ has a relaxed bounded constraint than $P^\lambda$, $\bar{W}$ may not be a feasible solution of $P^\lambda$. However, we can construct a feasible solution $\hat{W} = \bar{W} - G \circ E$, where $G_{ij} = \text{sign}(W_{ij})$ and $\circ$ indicates the entrywise product of two matrices. The assumption of this theorem implies that $\|G \circ E\|_2 \leq (1-\gamma)/\|\bar{W}\|_2$, so $\|\bar{W}^{-1}(G \circ E)\| \leq (1-\gamma)$. From Lemma 1 we have $\log \det \hat{W} \geq \log \det \bar{W} - \frac{p}{\gamma \sigma_{min}(\bar{W})}\|E\|_F$. Since $W^*$ is the optimal solution of $P^\lambda$ and $\hat{W}$ is a feasible solution of $P^\lambda$, $\log \det W^* \geq \log \det \hat{W} \geq \log \det \bar{W} - \frac{p}{\gamma \sigma_{min}(\bar{W})}\|E\|_F$. Also, since $\bar{W}$ is the optimal solution of $P^{\bar{\lambda}}$ and $W^*$ is a feasible solution of $P^{\bar{\lambda}}$, we have $\log \det W^* < \log \det \bar{W}$. Therefore, $|\log \det \bar{W} - \log \det W^*| < \frac{p}{\gamma \sigma_{min}(\bar{W})}\|E\|_F$.

By the mean value theorem and some calculations, we have $|f(W^*) - f(\bar{W})| > \frac{\|\bar{W} - W^*\|_F}{\max(\sigma_{max}(\bar{W}), \sigma_{max}(W^*))}$, which implies (7).

To establish the bound on $\Theta$, we use the mean value theorem again with $g(W) = W^{-1} = \Theta$, $\nabla g(W) = \Theta \otimes \Theta$ where $\otimes$ is kronecker product. Moreover, $\sigma_{max}(\Theta \otimes \Theta) = (\sigma_{max}(\Theta))^2$, so we can combine with (7) to prove (8). □

### 3.2 Clustering algorithm

In order to obtain computational savings, the clustering algorithm for the divide-and-conquer algorithm (Algorithm 1) should satisfy three conditions: (1) minimize the distance between the approximate and the true solution $\|\bar{\Theta} - \Theta^*\|_F$, (2) be cheap to compute, and (3) partition the nodes into balanced clusters.

Assume the real inverse covariance matrix $\Theta^*$ is block-diagonal, then it is easy to show that $W^*$ is also block-diagonal. This is the case considered in [8]. Now let us assume $\Theta^*$ has almost a block-diagonal structure but a few off-diagonal entries are not zero. Assume $\Theta^* = \Theta^{bd} + v e_i e_j^T$ where $\Theta^{bd}$ is the block-diagonal part of $\Theta^*$ and $e_i$ denotes the $i$-th standard basis vector, then from Sherman-Morrison formula,

$$W^* = (\Theta^*)^{-1} = (\Theta^{bd})^{-1} - \frac{v}{1 + v(\Theta^{bd})_{ij}} \theta_i^{bd}(\theta_j^{bd})^T,$$

where $\theta_i^{bd}$ is the $i$th column vector of $\Theta^{bd}$. Therefore adding one off-diagonal element to $\Theta^{bd}$ will introduce at most one nonzero off-diagonal block in $W$. Moreover, if block $(i, j)$ of $W$ is already nonzero, adding more elements in block $(i, j)$ of $\Theta$ will not introduce any more nonzero blocks in $W$. As long as just a few entries in off-diagonal blocks of $\Theta^*$ are nonzero, $W$ will be block-diagonal with a few nonzero off-diagonal blocks. Since $\|W^* - S^*\|_\infty \leq \lambda$, we are able to use the thresholding matrix $S^\lambda$ to guess the clustering structure of $\Theta^*$.

In the following, we show this observation is consistent with the bound we get in Theorem 3. From (8), ideally we want to find a partition to minimize $\|E\|_* = \sum_i |\sigma_i(E)|$. Since it is computationally difficult to optimize this directly, we can use the bound $\|E\|_* \leq \sqrt{p}\|E\|_F$, so that minimizing $\|E\|_F$ can be cast as a relaxation of the problem of minimizing $\|\bar{\Theta} - \Theta^*\|_F$.

To find a partition minimizing $\|E\|_F$, we want to find a partition $\{\mathcal{V}_c\}_{c=1}^k$ such that the sum of off-diagonal block entries of $S^\lambda$ is minimized, where $S^\lambda$ is defined as

$$(S^\lambda)_{ij} = \max(|S_{ij}| - \lambda, 0)^2 \; \forall \, i \neq j \;\; \text{and} \;\; S_{ij}^\lambda = 0 \; \forall i = j. \tag{10}$$

At the same time, we want to have balanced clusters. Therefore, we minimize the following normalized cut objective value [10]:

$$NCut(S^\lambda, \{\mathcal{V}_c\}_{c=1}^k) = \sum_{c=1}^k \frac{\sum_{i \in \mathcal{V}_c, j \notin \mathcal{V}_c} S_{ij}^\lambda}{d(\mathcal{V}_c)} \;\; \text{where} \;\; d(\mathcal{V}_c) = \sum_{i \in \mathcal{V}_c} \sum_{j=1}^p S_{ij}^\lambda. \tag{11}$$

In (11), $d(\mathcal{V}_c)$ is the volume of the vertex set $V_c$ for balancing cluster sizes, and the numerator is the sum of off-diagonal entries, which corresponds to $\|E\|_F^2$. As shown in [10, 3], minimizing the normalized cut is equivalent to finding cluster indicators $\boldsymbol{x}_1, \ldots, \boldsymbol{x}_c$ to maximize

$$\min_{\boldsymbol{x}} \sum_{c=1}^k \frac{\boldsymbol{x}_c^T (D - S^\lambda) \boldsymbol{x}_c}{\boldsymbol{x}_c^T D \boldsymbol{x}} = \text{trace}(Y^T (I - D^{-1/2} S^\lambda D^{-1/2}) Y), \tag{12}$$

where $D$ is a diagonal matrix with $D_{ii} = \sum_{j=1}^p S_{ij}^\lambda$, $Y = D^{1/2} X$ and $X = [\boldsymbol{x}_1 \ldots \boldsymbol{x}_c]$. Therefore, a common way for getting cluster indicators is to compute the leading $k$ eigenvectors of $D^{-1/2} S^\lambda D^{-1/2}$ and then conduct kmeans on these eigenvectors.

The time complexity of normalized cut on $S^\lambda$ is mainly from computing the leading $k$ eigenvectors of $D^{-1/2} S^\lambda D^{-1/2}$, which is at most $O(p^3)$. Since most state-of-the-art methods for solving (1) require $O(p^3)$ per iteration, the cost for clustering is no more than one iteration for the original solver. If $S^\lambda$ is sparse, as is common in real situations, we could speed up the clustering phase by using the Graclus multilevel algorithm, which is a faster heuristic to minimize normalized cut [3].

## 4 Experimental Results

In this section, we first show that the normalized cut criterion for the thresholded matrix $S^\lambda$ in (10) can capture the block diagonal structure of the inverse covariance matrix before solving (1). Using the clustering results, we show that our divide and conquer algorithm significantly reduces the time needed for solving the sparse inverse covariance estimation problem.

We use the following datasets:

1. Leukemia: Gene expression data — originally provided by [5], we use the data after the pre-processing done in [7].
2. Climate: This dataset is generated from NCEP/NCAR Reanalysis data [1], with focus on the daily temperature at several grid points on earth. We treat each grid point as a random variable, and use daily temperature in year 2001 as features.
3. Stock: Financial dataset downloaded from Yahoo Finance [2]. We collected 3724 stocks, each with daily closing price recorded in latest 300 days before May 15, 2012.
4. Synthetic: We generated synthetic data containing $20,000$ nodes with 100 randomly generated group centers $\mu_1, \ldots, \mu_{100}$, each of dimension 200, such that each group $c$ has half of its nodes with feature $\mu_c$ and the other half with features $-\mu_c$. We then add Gaussian noise to the features.

The data statistics are summarized in Table 1.

### 4.1 Clustering quality on real datasets

Given a clustering partition $\{\mathcal{V}_c\}_{c=1}^k$, we use the following "within-cluster ratio" to determine its performance on $\Theta^*$:

$$\mathcal{R}(\{\mathcal{V}_c\}_{c=1}^k) = \frac{\sum_{c=1}^k \sum_{i,j: i \neq j \text{ and } i,j \in \mathcal{V}_c} (\Theta_{ij}^*)^2}{\sum_{i \neq j} (\Theta_{ij}^*)^2}. \tag{13}$$

Table 1: Dataset Statistics

| | Leukemia | Climate | Stock | Synthetic |
|---|---|---|---|---|
| $p$ | 1255 | 10512 | 3724 | 20000 |
| $n$ | 72 | 1464 | 300 | 200 |

Table 2: Within-cluster ratios (see (13)) on real datasets. We can see that our proposed clustering method Spectral $S^\lambda$ is very close to the clustering based on $\hat{\Theta} = \Theta^* \circ \Theta^*$, which we cannot see before solving (1).

| | Leukemia | | Climate | | Stock | | Synthetic | |
|---|---|---|---|---|---|---|---|---|
| | $\lambda=0.5$ | $\lambda=0.3$ | $\lambda=0.005$ | $\lambda=0.001$ | $\lambda=0.0005$ | $\lambda=0.0001$ | $\lambda=0.005$ | $\lambda=0.001$ |
| random clustering | 0.26 | 0.24 | 0.24 | 0.25 | 0.24 | 0.24 | 0.25 | 0.24 |
| spectral on $S^\lambda$ | 0.91 | 0.84 | 0.87 | 0.65 | 0.96 | 0.87 | 0.98 | 0.93 |
| spectral on $\hat{\Theta}$ | 0.93 | 0.84 | 0.90 | 0.71 | 0.97 | 0.85 | 0.99 | 0.93 |

Higher values of $\mathcal{R}(\{\mathcal{V}_c\}_{c=1}^k)$ are indicative of better performance of the clustering algorithm.

In section 3.1, we presented theoretical justification for using normalized cut on the thresholded matrix $S^\lambda$. Here we show that this strategy shows great promise on real datasets. Table 2 shows the within-cluster ratios (13) of the inverse covariance matrix using different clustering methods. We include the following methods in our comparison:

- Random partition: partition the nodes randomly into $k$ clusters. We use this as a baseline.

- Spectral clustering on thresholded matrix $S^\lambda$: Our proposed method.

- Spectral clustering on $\hat{\Theta} = \Theta^* \circ \Theta^*$, which is the element-wise square of $\Theta^*$: This is the best clustering method we can conduct, which directly minimizes within-cluster ratio of the $\Theta^*$ matrix. However, practically we cannot use this method as we do not know $\Theta^*$.

We can observe in Table 2 that our proposed spectral clustering on $S^\lambda$ achieves almost the same performance as spectral clustering on $\Theta^* \circ \Theta^*$ even though we do not know $\Theta^*$.

Also, Figure 1 gives a pictorial view of how our clustering results help in recovering the sparse inverse covariance matrix at different levels. We run a hierarchical 2-way clustering on the Leukemia dataset, and plot the original $\Theta^*$ (solution of (1)), $\bar{\Theta}$ with 1-level clustering and $\bar{\Theta}$ with 2-level clustering. We can see that although our clustering method does not look at $\Theta^*$, the clustering result matches the nonzero pattern of $\Theta^*$ pretty well.

## 4.2 The performance of our divide and conquer algorithm

Next, we investigate the time taken by our divide and conquer algorithm on large real and synthetic datasets. We include the following methods in our comparisons:

- DC-QUIC-1: Divide and Conquer framework with QUIC and with 1 level clustering.

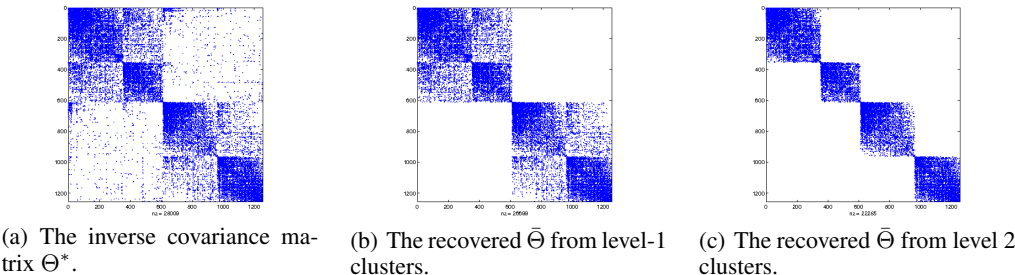

(a) The inverse covariance matrix $\Theta^*$.

(b) The recovered $\bar{\Theta}$ from level-1 clusters.

(c) The recovered $\bar{\Theta}$ from level 2 clusters.

Figure 1: The clustering results and the nonzero patterns of inverse covariance matrix $\Theta^*$ on Leukemia dataset. Although our clustering method does not look at $\Theta^*$, the clustering results match the nonzero pattern in $\Theta^*$ pretty well.

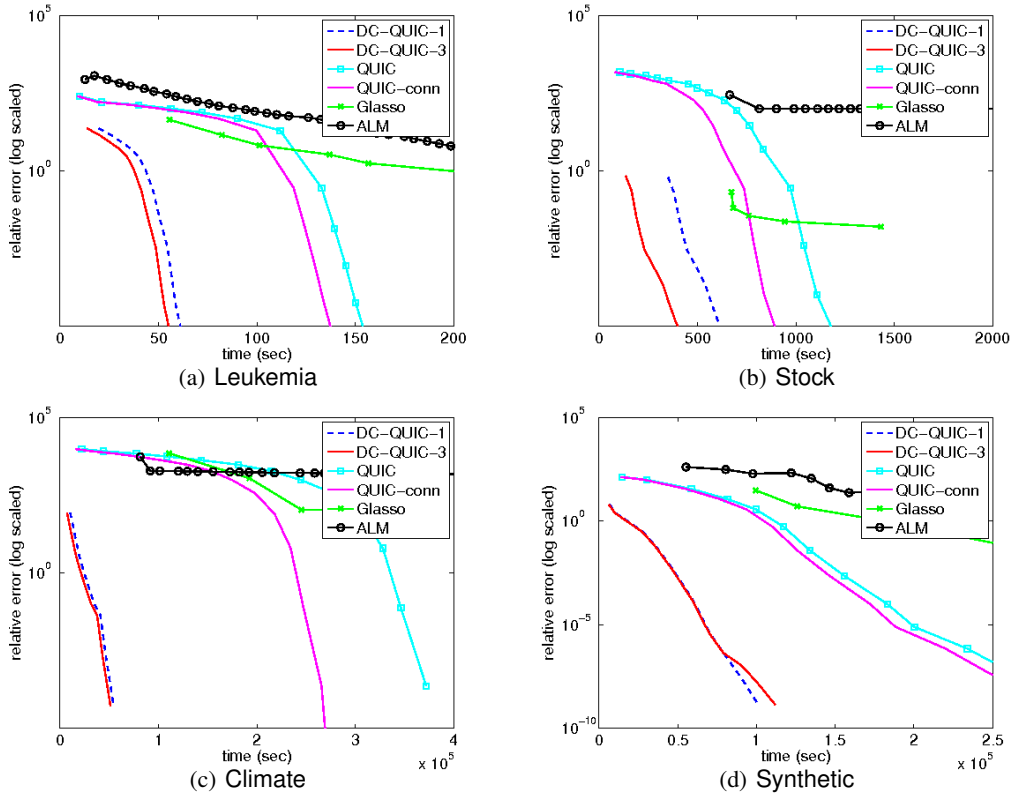

Figure 2: Comparison of algorithms on real datasets. The results show that DC-QUIC is much faster than other state-of-the-art solvers.

- DC-QUIC-3: Divide and Conquer QUIC with 3 levels of hierarchical clustering.
- QUIC: The original QUIC, which is a state-of-the-art second order solver for sparse inverse estimation [6].
- QUIC-conn: Using the decomposition method described in [8] and using QUIC to solve each smaller sub-problem.
- Glasso: The block coordinate descent algorithm proposed in [4].
- ALM: The alternating linearization algorithm proposed and implemented by [9].

All of our experiments are run on an Intel Xeon E5440 2.83GHz CPU with 32GB main memory. Figure 2 shows the results. For DC-QUIC and QUIC-conn, we show the run time of the whole process, including the preprocessing time. We can see that in the largest synthetic dataset, DC-QUIC is more than 10 times faster than QUIC, and thus also faster than Glasso and ALM. For the largest real dataset: Climate with more than 10,000 points, QUIC takes more than 10 hours to get a reasonable solution (relative error=0), while DC-QUIC-3 converges in 1 hour. Moreover, on these 4 datasets QUIC-conn using the decomposition method of [8] provides limited savings, in part because the connected components for the thresholded covariance matrix for each dataset turned out to have a giant component, and multiple smaller components. DC-QUIC however was able to leverage a reasonably good clustered decomposition, which dramatically reduced the inference time.

## Acknowledgements

We would like to thank Soumyadeep Chatterjee and Puja Das for help with the climate and stock data. C.-J.H., I.S.D and P.R. acknowledge the support of NSF under grant IIS-1018426. P.R. also acknowledges support from NSF IIS-1149803. A.B. acknowledges support from NSF grants IIS-0916750, IIS-0953274, and IIS-1029711.

## Footnotes

[1] www.esrl.noaa.gov/psd/data/gridded/data.ncep.reanalysis.surface.html

[2] http://finance.yahoo.com/

# References

[1] O. Banerjee, L. E. Ghaoui, and A. d'Aspremont. Model selection through sparse maximum likelihood estimation for multivariate Gaussian or binary data. *The Journal of Machine Learning Research*, 9, 6 2008.

[2] R. Bhatia. *Matrix Analysis*. Springer Verlag, New York, 1997.

[3] I. S. Dhillon, Y. Guan, and B. Kulis. Weighted graph cuts without eigenvectors: A multi-level approach. *IEEE Transactions on Pattern Analysis and Machine Intelligence (TPAMI)*, 29:11:1944–1957, 2007.

[4] J. Friedman, T. Hastie, and R. Tibshirani. Sparse inverse covariance estimation with the graphical lasso. *Biostatistics*, 9(3):432–441, July 2008.

[5] T. R. Golub, D. K. Slonim, P. Tamayo, C. Huard, M. Gaasenbeek, J. P. Mesirov, H. Coller, M. L. Loh, J. R. Downing, M. A. Caligiuri, and C. D. Bloomfield. Molecular classication of cancer: class discovery and class prediction by gene expression monitoring. *Science*, pages 531–537, 1999.

[6] C.-J. Hsieh, M. Sustik, I. S. Dhillon, and P. Ravikumar. Sparse inverse covariance matrix estimation using quadratic approximation. In *NIPS*, 2011.

[7] L. Li and K.-C. Toh. An inexact interior point method for l1-reguarlized sparse covariance selection. *Mathematical Programming Computation*, 2:291–315, 2010.

[8] R. Mazumder and T. Hastie. Exact covariance thresholding into connected components for large-scale graphical lasso. *Journal of Machine Learning Research*, 13:723–736, 2012.

[9] K. Scheinberg, S. Ma, and D. Glodfarb. Sparse inverse covariance selection via alternating linearization methods. *NIPS*, 2010.

[10] J. Shi and J. Malik. Normalized cuts and image segmentation. *IEEE Trans. Pattern Analysis and Machine Intelligence*, 22(8):888–905, 2000.

